# A Bayesian Approach to Diffusion Models of Decision-Making and Response Time

**Michael D. Lee**[*]
Department of Cognitive Sciences
University of California, Irvine
Irvine, CA, 92697-5100.
mdlee@uci.edu

**Ian G. Fuss**
Defence Science and Technology Organisation
PO Box 1500, Edinburgh, SA 5111, Australia
ian.fuss@dsto.defence.gov.au

**Daniel J. Navarro**
School of Psychology
University of Adelaide, SA 5005, Australia
daniel.navarro@adelaide.edu.au

## Abstract

We present a computational Bayesian approach for Wiener diffusion models, which are prominent accounts of response time distributions in decision-making. We first develop a general closed-form analytic approximation to the response time distributions for one-dimensional diffusion processes, and derive the required Wiener diffusion as a special case. We use this result to undertake Bayesian modeling of benchmark data, using posterior sampling to draw inferences about the interesting psychological parameters. With the aid of the benchmark data, we show the Bayesian account has several advantages, including dealing naturally with the parameter variation needed to account for some key features of the data, and providing quantitative measures to guide decisions about model construction.

## 1   Introduction

In the past decade, modern computational Bayesian methods have been productively applied to the modeling of many core psychological phenomena. These areas include similarity modeling and structure learning [1], concept and category learning [2, 3], inductive inference and decision-making [4], language processes [5], and individual differences [6]. One central area that has been less affected is the modeling of response times in decision-making.

Nevertheless, the time people take to produce behavior is a basic and ubiquitious measure that can constrain models and theories of human cognitive processes [7]. There is a large and well-developed set of competing models that aim to account for accuracy, response time distributions and (sometimes) confidence in decision-making. However, besides the effective application of hierarchical Bayesian methods to models that assume response times follow a Weibull distribution [8], most of the inference remains frequentist. In particular, sequential sampling models of response time, which are the dominant class in the field, have not adopted modern Bayesian methods for inference. The prominent recent review paper by Ratcliff and Smith, for example, relies entirely on frequentist methods for parameter estimation, and does not go beyond the application of the Bayesian Information Criterion for model selection [9].

---
[*]Address correspondence to: Michael D. Lee, Department of Cognitive Sciences, 3151 Social Sciences Plaza, University of California, Irvine, CA 92697-5100. Telephone: (949) 824 5074. Facsimile: (949) 824 2307. URL: www.socsci.uci.edu/∼mdlee.

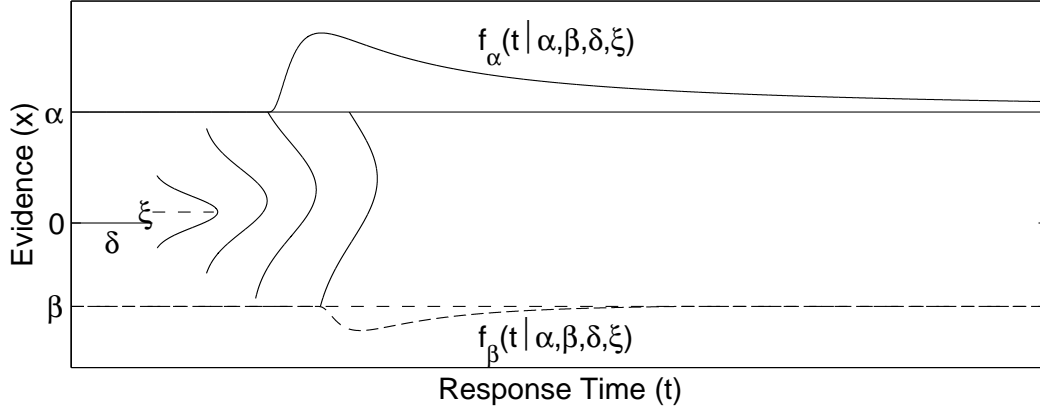

Figure 1: A diffusion model for response time distributions for both decisions in a two-choice decision-making task. See text for details.

Much of the utility, however, in using sequential sampling models to understand decision-making, and their application to practical problems [10], requires making inferences about variations in parameters across subjects, stimuli, or experimental conditions. These inferences would benefit from the principled representation of uncertainty inherent in the Bayesian approach. In addition, many of the competing models have many parameters, are non-linear, and are non-nested. This means their comparison would benefit from Bayesian methods for model selection that do not approximate model complexity by counting the number of free parameters, as the Bayesian Information Criterion does.

In this paper, we present a computational Bayesian approach for Wiener diffusion models [11], which are the most widely used special case of the sequential sampling approach. We apply our Bayesian method to the benchmark data of Ratcliff and Rouder [12], using posterior sampling to draw inferences about the interesting psychological parameters. With the aid of this application, we show that adopting the Bayesian perspective has several advantages, including dealing naturally with the parameter variation needed to account for some key features of the data, and providing quantitative measures to guide decisions about model construction.

## 2 The Diffusion Model and its Application to Benchmark Data

### 2.1 The Basic Model

The basic one-dimensional diffusion model for accuracy and response time distributions in a two-choice decision-making task is shown in Figure 2.1. Time, $t$, progresses from left to right, and includes an fixed offset $\delta$ that parameterizes the time taken for the non-decision component of response time, such as the time taken to encode the stimulus and complete a motor response.

The decision-making component itself is driven by independent samples from an stationary distribution that represents the evidence the stimulus provides in favor of the two alternative decisions. In the Wiener diffusion, this distribution is assumed to be Gaussian, with mean $\xi$. Evidence sampled from this distribution is accrued over time, leading to a diffusion process that is finally absorbed by boundaries above and below at distances $\alpha$ and $\beta$ from the origin. The response time distribution is then given by the first-passage distribution $p\left(t \mid \alpha, \beta, \delta, \xi\right) = f_\alpha\left(t \mid \alpha, \beta, \delta, \xi\right) + f_\beta\left(t \mid \alpha, \beta, \delta, \xi\right)$, with the areas under $f_\alpha$ and $f_\beta$ giving the proportion of decisions at each boundary.

A natural reparameterization is to consider the starting point of evidence accrual $z = (\beta - \alpha) / (\alpha + \beta)$, which is considered a measure of bias, and the boundary separation $a = \alpha + \beta$, which is considered a measure of caution. In either case, this basic form of the model has four free parameters: $\xi$, $\delta$ and either $\alpha$ and $\beta$ or $z$ and $a$.

## 2.2 Previous Application to Benchmark Data

The evolution of Wiener diffusion models of decision-making has involved a series of additional assumptions to address shortcomings in its ability to capture basic empirical regularities. This evolution is well described by Ratcliff and Rouder [12], who, in their Experiment 1 present a diffusion model analysis of a benchmark data set [8, 13].

In this experiment, three observers completed ten 35 minute sessions, each consisting of ten blocks with 102 trials per block. The task of observers was to decide between 'bright' and 'dark' responses for simple visual stimuli with different proportions of white dots, given noisy feedback about the accuracy of the responses. There were 33 different types of stimuli, ranging from 0% to 100% white in equal increments. In addition, the subjects were required to switch between adherence to 'speed' instructions and 'accuracy' instructions every two blocks. In accord with the experimental design, Ratcliff and Rouder fitted separate drift rates $\xi_i$ for each of the $i = 1, \ldots, 33$ stimuli; fitted separate boundaries for speed and accuracy instructions, but assumed the boundaries were symmetric (i.e., there was no bias), so that $\alpha_j = \beta_j$ for $j = 1, 2$; and fitted one offset $\delta$ for all stimuli and instructions.

The data from this experiment are considered benchmark because they show a cross-over effect, whereby errors are faster than correct decisions for easy stimulus conditions under speed instructions, but errors are as slow or slower than correct decisions for hard stimulus conditions under accuracy instructions. As Ratcliff and Rouder point out, these trends are not accommodated by the basic model without allowing for variation in the parameters. Accordingly, to predict fast errors, the basic model is extended by assuming that the starting point is subject to between-trial variation, and so is convolved with a Gaussian or uniform distribution. Similarly, to predict slow errors, it is assumed that the mean drift rate is also subject to between-trial variation, and so is convolved with a Gaussian distribution. Both of these noise processes are parameterized with the standard sufficient statistics, which become additional parameters of the model.

## 3 Closed-form Response Time Distributions for Diffusion Models

One practical reason diffusion models have resisted a Bayesian treatment is that the evaluation of their likelihood function through standard methods is computationally intensive, typically requiring the estimation of an oscillating but convergent infinite sum for each datum [14]. Instead, we use a new closed-form approximation to the required response time distribution. We give a very brief presentation of the approximation here. A more detailed technical note is available from the first author's web page.

The key assumption in our approximation is that the evolving diffusion distributions always assume a limiting form $f$. Given this form, we define the required limit for a sampling distribution with respect to an arbitrary time dependent mean $\mu(t, \boldsymbol{\theta})$ and variance $\sigma^2(t, \boldsymbol{\theta})$, both which depend on parameters $\boldsymbol{\theta}$ of the sampling distribution, in terms of the evidence accumulated $x$, as

$$f\left(x; \mu(t, \boldsymbol{\theta}), \sigma^2(t, \boldsymbol{\theta})\right), \tag{1}$$

from which the cumulative function at an upper boundary of one unit is obtained as

$$F_1\left(\mu(t, \boldsymbol{\theta}), \sigma^2(t, \boldsymbol{\theta})\right) = \int_1^\infty f\left(x; \mu(t, \boldsymbol{\theta}), \sigma^2(t, \boldsymbol{\theta})\right) \, \mathrm{d}x. \tag{2}$$

Differentiation, followed by algebraic manipulation gives the general result

$$
\begin{aligned}
f_1\left(\mu(t, \boldsymbol{\theta}), \sigma^2(t, \boldsymbol{\theta})\right) &= \frac{\mathrm{d}}{\mathrm{d}t} F_1\left(\mu(t, \boldsymbol{\theta}), \sigma^2(t, \boldsymbol{\theta})\right) \\
&= \frac{\frac{\partial}{\partial t}\sigma(t, \boldsymbol{\theta})\left[1 - \mu(t, \boldsymbol{\theta})\right] + \sigma(t, \boldsymbol{\theta})\frac{\partial}{\partial t}\mu(t, \boldsymbol{\theta})}{\sigma^2(t, \boldsymbol{\theta})} f\left(\frac{1 - \mu(t, \boldsymbol{\theta})}{\sigma(t, \boldsymbol{\theta})}\right).
\end{aligned}
\tag{3}
$$

Weiner diffusion from the origin to boundaries $\alpha$ and $\beta$ with mean drift rate $\xi$ and variance $\sigma^2(t) = t$ can be represented in this model by defining $f(y) = \exp\left(-\frac{1}{2}y^2\right)$, rescaling to a variance $\sigma^2(t) = \frac{1}{a^2}t$ and setting $\mu(t, \xi) = \frac{1}{a}\xi t + z$. Thus the response time distributions for Weiner diffusion in this

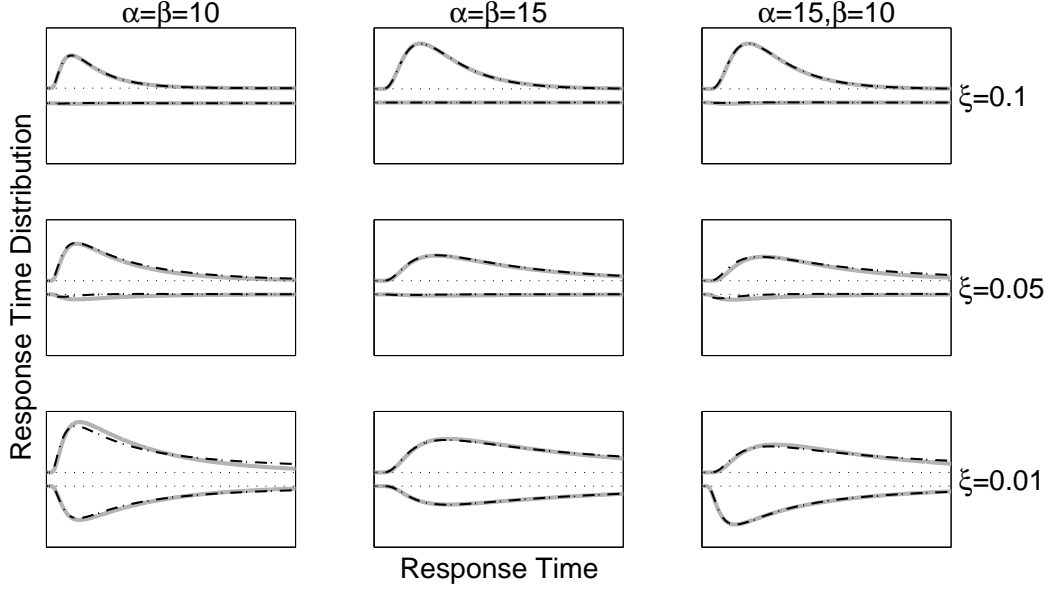

Figure 2: Comparison between the closed form approximation (dark broken lines) and the infinite sum distributions (light solid lines) for nine realistic combinations of drift rate and boundaries.

approximation are

$$
\begin{aligned}
f_\alpha\left(t \mid \alpha, \beta, \delta, \xi\right) &= \frac{2\alpha + \xi\left(t - \delta\right)}{2\left(t - \delta\right)^{\frac{3}{2}}} \exp\left(-\frac{\left(2\alpha - \xi\left(t - \delta\right)\right)^2}{2\left(t - \delta\right)}\right), \\
f_\beta\left(t \mid \alpha, \beta, \delta, \xi\right) &= \frac{2\beta - \xi\left(t - \delta\right)}{2\left(t - \delta\right)^{\frac{3}{2}}} \exp\left(-\frac{\left(2\beta + \xi\left(t - \delta\right)\right)^2}{2\left(t - \delta\right)}\right).
\end{aligned}
\tag{4}
$$

### 3.1 Adequacy of the Wiener Approximation

Figure 2 shows the relationship between the response time distributions found by the previous infinite sum method, and those generated by our closed form approximation. For every combination of drift rates $\xi = 0.01$, $0.05$ and $0.10$, and boundary combinations $\alpha = \beta = 10$, $\alpha = \beta = 15$ and $\alpha = 15$, $\beta = 10$ we found the best (least-squares) match between the infinite-sum distribution and those distributions indexed by our approximation. These generating parameter combinations were chosen because they cover the range of the posterior distributions we infer from data later. Figure 2 shows that the approximation provides close matches across these parameterizations, although we note the approximation distributions do seem to use slightly (and apparently systematically) different parameter combinations to generate the best-matching distribution. While additional work is required to understand the exact relationship between the infinite-sum method and our approximation, the approximation is sufficiently accurate over the range of parameterizations of interest to be used as the basis for beginning to apply Bayesian methods to diffusion models.

## 4 Bayesian Modeling of Benchmark Data

### 4.1 General Model

Our log likelihood function evaluates the density of each response time at the boundary corresponding to its associated decision, and assumes independence, so that

$$
\ln \mathcal{L}\left(T \mid \alpha, \beta, \delta, \xi\right) = \sum_{t \in D_\alpha} \ln f_\alpha\left(t \mid \alpha, \beta, \delta, \xi\right) + \sum_{t \in D_\beta} \ln f_\beta\left(t \mid \alpha, \beta, \delta, \xi\right),
\tag{5}
$$

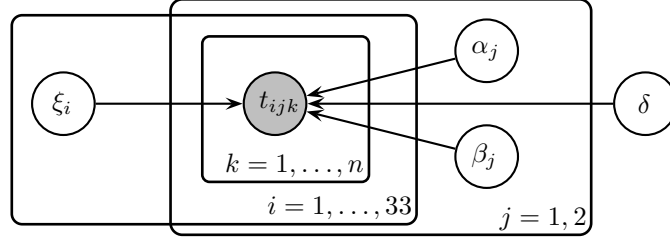

Figure 3: Graphical model for the benchmark data analysis.

where $D_\alpha$ and $D_\beta$ are the set of all response times at the upper and lower boundaries respectively. We threshold at $10^{-30}$ to guard against degeneracy to negative values inherent in the approximation.

The graphical model representation of the benchmark data is shown in Figure 3, where the observed response time data are now denoted $T = \{t_{ijk}\}$, with $i = 1, \ldots, 33$ indexing the presented stimulus, $j = 1, 2$ indexing speed or accuracy instructions, and $k = 1, \ldots, n$ indexing all of the trials with this stimulus and instruction combination. We place proper approximations to non-informative distributions on all the parameters, so that they are all essentially flat over the values of interest. Specifically we assume the 33 drift rates are independent and each have a zero mean Gaussian prior with very small precision: $\xi_i \sim \mathrm{Gaussian}(0, \tau)$, with $\tau = 10^{-6}$. The boundary parameters $\alpha$ and $\beta$ are given the same priors, but because they are constrained to be positive, their sampling is censored accordingly: $\alpha_j, \beta_j \sim \mathrm{Gaussian}(0, \tau)$; $\alpha_j, \beta_j > 0$. Since $\delta$ is bounded by the minimum time observed, we use the Uniform prior distribution: $\delta \sim \mathrm{Uniform}(0, \min T)$. This is a data-dependent prior, but the same results could be achieved with a fixed prior, and scaling of the time data, which are arbitrary up to scalar multiplication.

## 4.2 Formalizing Model Construction

The Bayesian approach allows us to test the intuitively plausible model construction decisions made previously by Ratcliff and Rouder. Using the data from one of the observers (N.H.), we considered the marginal likelihoods, denoted simply $\mathcal{L}$, based on the harmonic mean approximation [15], calculated from three chains of $10^5$ samples from the posterior obtained using Winbugs.

- The full model described by Figure 3, with asymmetric boundaries, varying across speed and accuracy instructions. This model had $\ln \mathcal{L} = -48, 416$.

- The restricted model with symmetric boundaries, still varying across instructions, as assumed by Ratcliff and Rouder. This model had $\ln \mathcal{L} = -48, 264$.

- The restricted model with asymmetric boundaries not varying across instructions. This model had $\ln \mathcal{L} = -48, 964$.

- The restricted model with symmetric boundaries not varying across instructions. This model had $\ln \mathcal{L} = -48, 907$.

These marginal log likelihoods make it clear that different boundaries are needed for the speed and accuracy instructions, but it is overly complicated to allow them to be asymmetric (i.e., there is no need to parameterize bias). We tested the robustness of these values by halving and doubling the prior variances, and using an adapted form of the 'informative' priors collated in [16], all of which lead to similar quantitative and identical qualitative conclusions. These results formally justify the model construction decisions made by Ratcliff and Rouder, and the remainder of our analysis applies to this restricted model.

## 4.3 Posterior Distributions

Figure 4 shows the posterior distributions for the symmetric boundaries under both speed and accuracy instructions. These distributions are consistent with traditional analyses and the speed boundary is clearly significantly smaller than the accuracy boundary. We note that, for historical reasons only, Wiener diffusion models have assumed $\sigma^2(t) = (0.1t)^2$, and so our scale is 100 times larger.

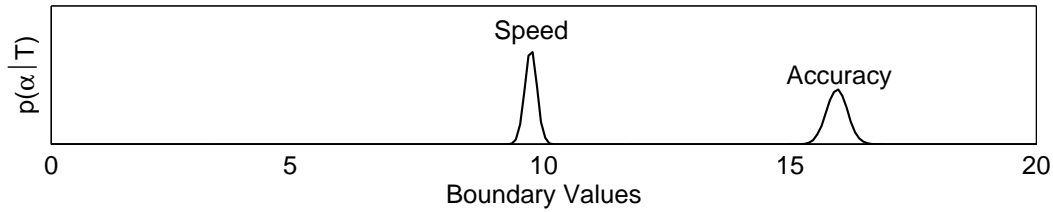

Figure 4: Posterior distributions for the boundaries under speed and accuracy instructions.

The main panel of Figure 5 shows the posterior distributions for all 33 drift rate parameters. The posteriors are shown against the vertical axes, with wider bars corresponding to greater density, and are located according to their proportion of white dots on the horizontal axis. The approximately monotonic relationship between drift rate and proportion shows that the model allows stimulus properties to be inferred from the behavioral decision time data, as found by previous analyses.

The right hand panel of Figure 5 shows the projection of three of the posterior distributions, labelled 4, 17 and 22. It is interesting to note that the uncertainty about the drift rate of stimuli 4 and 17 both take a Gaussian form, but with very different variances. More dramatically, the uncertainty about the drift rate for stimulus 22 is clearly bi-modal.

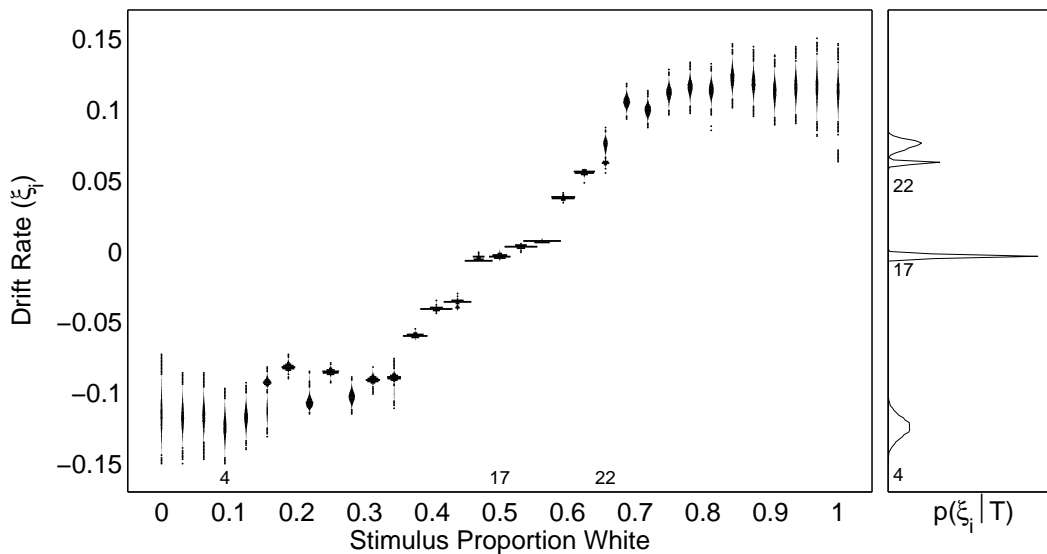

Figure 5: Posterior distributions for the 33 drift rates, in terms of the proportion of white dots in their associated stimuli.

## 4.4 Accuracy and Fast and Slow Errors

Figure 6 follows previous analyses of these data, and shows the relationship between the empirical proportions, and the model prediction of decision proportions for each stimulus type. For both the speed and accuracy instructions, there is close agreement between the model and data.

Figure 7 shows the posterior predictive distribution of the model for two cases, analogous to those highlighted previously [13, Figure 6]. The left panel involves a relatively easy decision, corresponding to stimulus number 22 under speed instructions, and shows the models predictions for the response time for both correct (upper) and error (lower) decisions, together with the data, indicated by short vertical lines. For this easy decision, it can be seen the model predicts relatively fast errors. The right panel of Figure 7 involves a harder decision, corresponding to stimulus number 18 under

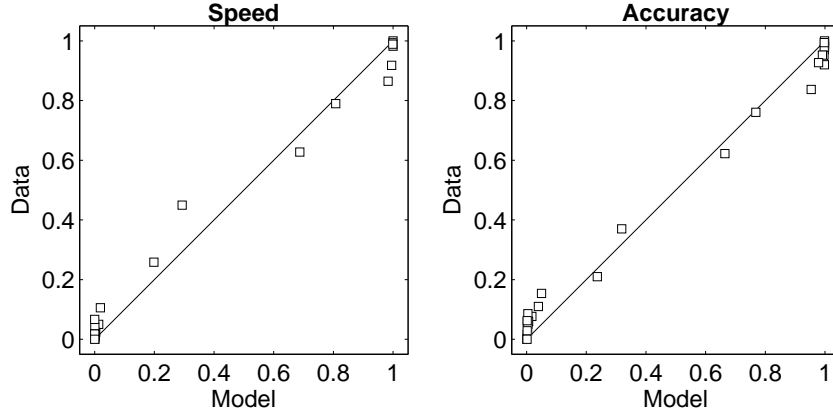

Figure 6: Relationship between modeled and empirical accuracy, for the speed instructions (left panel) and accuracy instructions (right panel). Each marker corresponds to one of the 33 stimuli.

accuracy instructions. Here the model predicts much slower errors, with a heavier tail than for the easy decision.

These are the basic qualitative properties of prediction that motivated the introduction of between-trial variability through noise processes in the traditional account. In the present Bayesian treatment, the required predictions are achieved because the posterior predictive automatically samples from a range of values for the drift and boundary parameters. By representing this variation in parameters as uncertainty about fixed values, we are making different basic assumptions from the traditional Wiener diffusion model. It is interesting to speculate that, if Bayesian results like those in Figure 7 had always been available, the introduction of additional variability processes described in [12] might never have eventuated. These processes seem solely designed to account for empirical effects like the cross-over effect; in particular, we are not aware of the parameters of the additional variability processes being used to draw substantive psychological inferences from data.

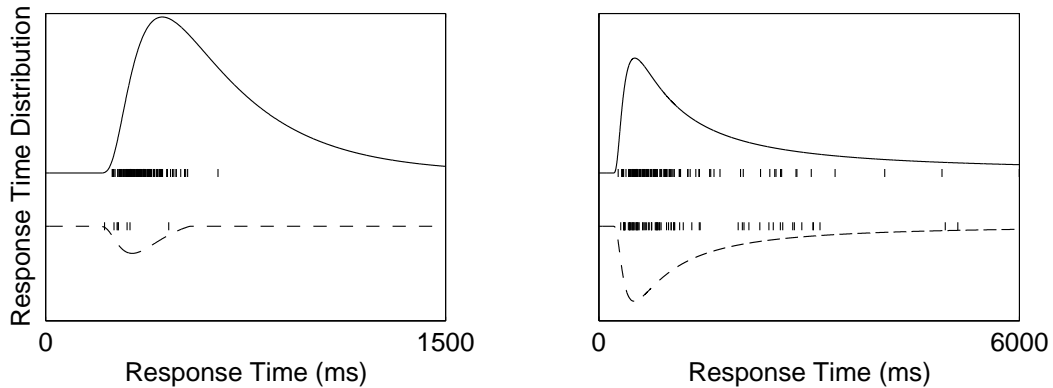

Figure 7: Posterior predictive distributions for both correct (solid line) and error (broken line) responses, for two stimuli corresponding to easy (left panel) and hard (right panel) decisions. The density for error decisions in the easy responses has been scaled to allow its shape to be visible.

## 5 Conclusions

Our analyses of the benchmark data confirm many of the central conclusions of previous analyses, but also make several new contributions. The posterior distributions shown in Figure 5 suggest that current parametric assumptions about drift rate variability may not be entirely appropriate. In

particular, there is the intriguing possibility of multi-modalities evident in the drift rate of stimulus 22, and the associated raw data in Figure 7.

Figure 5 also suggests a hierarchical account of the benchmark data, modeling the 33 drift rates $\xi_i$ in terms of, for example, a low-dimensional psychometric function. This would be easily achieved in the current Bayesian framework. It should also be possible to introduce contaminant distributions in a mixture model, following previous suggestions [8, 14], using latent variable assignments for each response time. If it was desirable to replicate the current assumptions of starting-point and drift-rate variability, that would also easily be done in an extended hierarchical account. Finally, the availability of marginal likelihood measures, accounting for both model fit and complexity, offer the possibility of rigorous quantitative comparisons of alternative sequential sampling accounts, such as the Ornstein-Uhlenbeck and accumulator models [9], of response times in decision-making.

**Acknowledgments**

We thank Jeff Rouder for supplying the benchmark data, and Scott Brown, E.-J. Wagenmakers, and the reviewers for helpful comments.

## References

[1] C. Kemp, A. Bernstein, and J. B. Tenenbaum. A generative theory of similarity. In B. G. Bara, L. W. Barsalou, and M. Bucciarelli, editors, *Proceedings of the 27th Annual Conference of the Cognitive Science Society*. Erlbaum, Mahwah, NJ, 2005.

[2] J. R. Anderson. The adaptive nature of human categorization. *Psychological Review*, 98(3):409–429, 1991.

[3] J. B. Tenenbaum and T. L. Griffiths. Generalization, similarity, and Bayesian inference. *Behavioral and Brain Sciences*, 24(4):629–640, 2001.

[4] T. L. Griffiths and J. B. Tenenbaum. Structure and strength in causal induction. *Cognitive Psychology*, 51:354–384, 2005.

[5] T. L. Griffiths, M. Steyvers, and D. Blei, and J. B. Tenenbaum. Integrating topics and syntax. *Advances in Neural Information Processing Systems*, 17, 2005.

[6] D. J. Navarro, T. L. Griffiths, M. Steyvers, and M. D. Lee. Modeling individual differences using Dirichlet processes. *Journal of Mathematical Psychology*, 50:101–122, 2006.

[7] R. D. Luce. *Response Times: Their Role in Inferring Elementary Mental Organization*. Oxford University Press, New York, 1986.

[8] J. N. Rouder, J. Lu, P. L. Speckman, D. Sun, and Y. Jiang. A hierarchical model for estimating response time distributions. *Psychonomic Bulletin & Review*, 12:195–223, 2005.

[9] R. Ratcliff and P. L. Smith. A comparison of sequential sampling models for two–choice reaction time. *Psychological Review*, 111:333–367, 2004.

[10] R. Ratcliff, A. Thapar, and G. McKoon. The effects of aging on reaction time in a signal detection task. *Psychology and Aging*, 16:323–341, 2001.

[11] R. Ratcliff. A theory of memory retrieval. *Psychological Review*, 85:59–108, 1978.

[12] R. Ratcliff and J. N. Rouder. Modeling response times for two–choice decisions. *Psychological Science*, 9:347–356, 1998.

[13] S. Brown and A. Heathcote. A ballistic model of choice response time. *Psychological Review*, 112:117–128, 1 2005.

[14] R. Ratcliff and F. Tuerlinckx. Estimating parameters of the diffusion model: Approaches to dealing with contaminant reaction times and parameter variability. *Psychonomic Bulletin & Review*, 9:438–481, 2002.

[15] A. E. Raftery. Hypothesis testing and model selection. In W. R. Gilks, S. Richardson, and D. J. Spiegelhalter, editors, *Markov chain Monte Carlo in Practice*, pages 163–187. Chapman & Hall/CRC, Boca Raton (FL), 1996.

[16] E.-J. Wagenmakers, H. J. L. van der Maas, and R. P. P. P. Grasman. An EZ–diffusion model for response time and accuracy. *Psychonomic Bulletin & Review*, in press.
